# A Self-organizing Associative Memory System for Control Applications

Michael Hormel
Department of Control Theory and Robotics
Technical University of Darmstadt
Schlossgraben 1
6100 Darmstadt/W.-Germany

## ABSTRACT

The CMAC storage scheme has been used as a basis for a software implementation of an associative memory system AMS, which itself is a major part of the learning control loop LERNAS. A major disadvantage of this CMAC-concept is that the degree of local generalization (area of interpolation) is fixed. This paper deals with an algorithm for self-organizing variable generalization for the AMS, based on ideas of T. Kohonen.

## 1 INTRODUCTION

For several years research at the Department of Control Theory and Robotics at the Technical University of Darmstadt has been concerned with the design of a learning real-time control loop with neuron-like associative memories (LERNAS)

for the control of unknown, nonlinear processes (Ersue, Tolle, 1988). This control concept uses an associative memory system AMS, based on the cerebellar cortex model CMAC by Albus (Albus, 1972), for the storage of a predictive nonlinear process model and an appropriate nonlinear control strategy (Fig. 1).

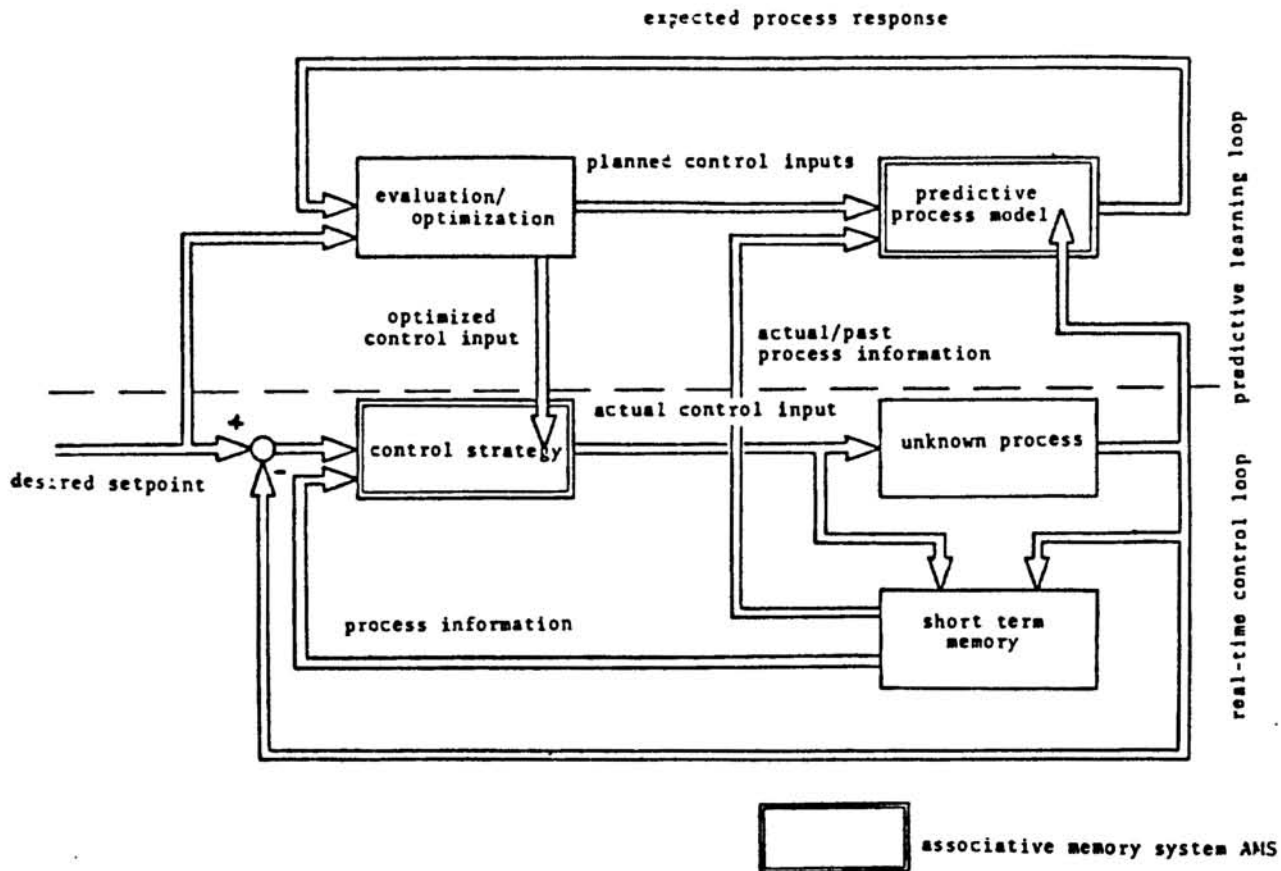

Figure 1: The learning control loop LERNAS

One problem for adjusting the control loop to a process is, however, to find a suitable set of parameters for the associative memory. The parameters in question determine the degree of generalization within the memory and therefore have a direct influence on the number of training steps required to learn the process behaviour. For a good performance of the control loop it is desirable to have a very small generalization around a given setpoint but to have a large generalization elsewhere. Actually, the amount of collected data is small during the transition phase between two

setpoints but is large during setpoint control. Therefore a self-organizing variable generalization, adapting itself to the amount of available data would be very advantageous.
Up to now, when working with fixed generalization, finding the right parameters has meant to find the best compromise between performance and learning time required to generate a process model. This paper will show a possibility to intro-duce a self-organizing variable generalization capability into the existing AMS/CMAC algorithm.

## 2 THE AMS-CONCEPT

The associative memory system AMS is based on the "Cerebel-lar Model Articulation Controller CMAC" as presented by J.S. Albus. The information processing structure of AMS can be divided into three stages.

1.) Each component of a n-dimensional input vector (stimu-lus) activates a fixed number $\rho$ of sensory cells, the receptive fields of which are overlapping. So $n \cdot \rho$ sen-sory cells become active.

2.) The active sensory cells are grouped to form $\rho$ n-dimen-sional vectors. These vectors are mapped to $\rho$ associa-tion cells. The merged receptive fields of the sensory cells described by one vector can be seen as a hypercube in the n-dimensional input space and therefore as the receptive field of the association cell. In normal ap-plications the total number of available association cells is about $100 \cdot \rho$.

3.) The association cells are connected to the output cells by modifiable synaptic weights. The output cell computes the mean value of all weights that are connected to ac-tive association cells (active weights).

Figure 2 shows the basic principle of the associative memory system AMS.

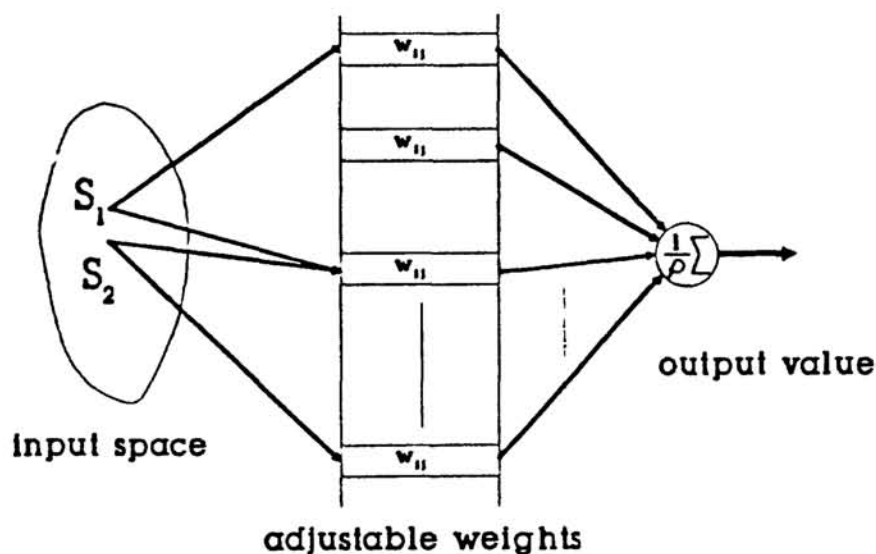

Figure 2: The basic mechanism of AMS

During training the generated output is compared with a desired output, the error is computed and equally distributed over all active weights. For the mapping of sensory cells to association cells a hash-coding mechanism is used.

# 3 THE SELF-ORGANIZING FEATURE MAP

An approach for explaining the self-organizing capabilities of the nervous system has been presented by T. Kohonen (Kohonen, 1988).

In his "self-organizing feature map" a network of laterally interconnected neurons can adapt itself according to the density of trained points in the input space. Presenting a n-dimensional input vector to the network causes every neuron to produce an output signal which is correlated with the similarity between the input vector and a "template vector" which may be stored in the synaptic weights of the neuron. Due to the "mexican-hat" coupling function between the neurons, the one with the maximum output activity will excite its nearest neighbours but will inhibit neurons farther away, therefore generating a localized response in the network. The active cells can now adapt their input weights in order to increase their similarity to the input vector. If we define the receptive field of a neuron by the number of input vectors for which the neurons activity is greater than

that of any other neuron in the net, this yields the effect
that in areas with a high density of trained points the re-
ceptive fields become small whereas in areas with a low den-
sity of trained points the size of the receptive fields is
large. As mentioned above this is a desired effect when
working with a learning control loop.

## 4 SELF-ORGANIZING VARIABLE GENERALIZATION

Both of the approaches above have several advantages and
disadvantages when using them for real-time control applica-
tions.

In the AMS algorithm one does not have to care for prede-
fining a network and the coupling functions or coupling ma-
trices among the elements of the network. Association and
weight cells are generated when they are needed during
training and can be adressed very quickly to produce a memo-
ry response. One of the disadvantages is the fixed generali-
zation once the parameters of a memory unit have been
chosen.

Unlike AMS, the feature map allows the adaption of the net-
work according to the input data. This advantage has to be
payed for by extensive search for the best matching neuron
in the network and therefore the response time of the net-
work may be too large for real-time control when working
with big networks.

These problems can be overcome when allowing that the map-
ping of sensory cells to association cells in AMS is no
longer fixed but can be changed during training.

To accomplish this a template vector $\underline{t}$ is introduced for
every association cell. This vector $\underline{t}$ serves as an indicator
for the stimuli by which the association cell has been ac-
cessed previously. During an associative recall for a stimu-
lus $\underline{s}_0$ a preliminary set of $\rho$ association cells is activated
by the hash coding mechanism. Due to the self-organizing
process during training the template vectors do not need to
correspond to the input vector $\underline{s}_0$. For the search for the

best matching cell the template vector $\underline{t}_0$ of the accessed association cell is compared to the stimulus and a difference vector is calculated.

$$\underline{\delta}_i = \underline{t}_i - \underline{s}_0 \qquad\qquad i = 0,\ldots,n_s \qquad\qquad (1)$$

$n_s$    number of searching steps

This vector can now be used to compute a virtual stimulus which compensates the mapping errors of the hash-coding mechanism.

$$\underline{s}_{i+1} = \underline{s}_i - \underline{\delta}_i \qquad\qquad i = 0,\ldots,n_s \qquad\qquad (2)$$

The best matching cell is found for

$$j = \min_i \| \underline{\delta}_i \| \qquad\qquad i = 0,\ldots,n_s \qquad\qquad (3)$$

and can be adressed by the virtual stimulus $\underline{s}_j$ when using the hash coding mechanism. This search mechanism ensures that the best matching cell is found even if self organization is in effect.

During training the template vectors of the association cells are updated by

$$\underline{t}(k+1) = \alpha(k,d) \cdot (\underline{s}(k) - \underline{t}(k)) + \underline{t}(k) \qquad\qquad (4)$$

d    lateral distance of neurons in the network

where $\underline{t}(k)$ denotes the value of the template vector at time k and $\underline{s}(k)$ denotes the stimulus. $\alpha(k,d)$ is a monotonic decreasing function of time and the lateral distance between neurons in the network.

# 5 SIMULATION RESULTS

Figure 3 and 4 show some simulation results of the presented algorithm for the dase of a two dimensional stimulus vector.

Figure 3 shows the expected positions in input space of the untrained template vectors ( x denotes untrained association cells).

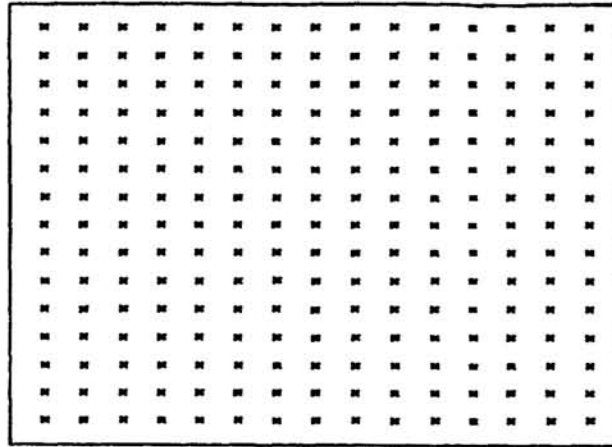

**Figure 3: Untrained Network**

Figure 4 shows the network after 2000 training steps with stimuli of gaussian distribution in input space. The position of the template vectors of trained cells has shifted into the direction of the better trained areas, so that more association cells are used to represent this area than before. Therefore the stored information will be more exact in this area.

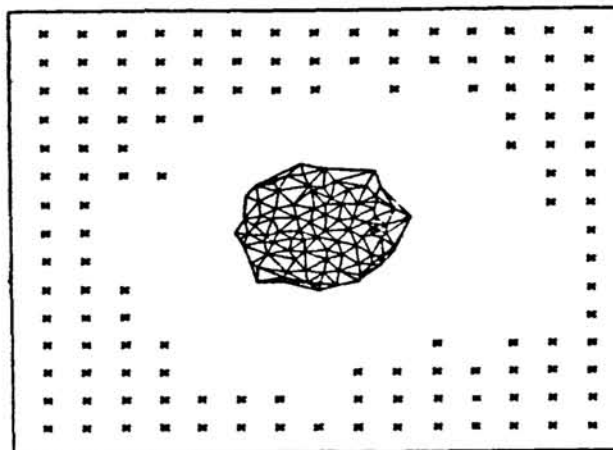

**Figure 4: Network after 2000 training steps**

# 6 CONCLUSION

The new algorithm presented above introduces the capability to adapt the storage mechanisms of a CMAC-type associative memory according to the arriving stimuli. This will result in various degrees of generalization depending on the number of trained points in a given area. It therefore will make it unnecessary to choose a generalization factor as a compromise between several constraints when representing nonlinear functions by storing them in this type of associative memory. Some results on tests will be presented together with a comparison on respective results for the original AMS.

**Acknowledgements**

This work was sponsored by the German Ministry for Research and Technology (BMFT) under grant no. ITR 8800 B/5

**References**

E. Ersue, H. Tolle. (1988) *Learning Control Structures with Neuron-Like Associative memories*. In: v. Seelen, Shaw, Leinhos (Eds.) Organization of Neural Networks, VCH Verlagsgesellschaft, Weinheim, FRG, 1988

J.S. Albus (1972) *Theoretical and experimental aspects of a cerebellar model*, PhD thesis, University of Maryland, USA

E. Ersue, X. Mao (1983) *Control of pH by Use of a Self-organizing Concept with Associative Memories*. ACI'83, Kopenhagen, Denmark

E. Ersue, J. Militzer (1984) *Real-time Implementation of an Associative Memory-based Learning Control Scheme for Nonlinear Multivariable Systems*. Symposium on "Applications of Multivariable System Techniques", Plymouth, UK

T. Kohonen. (1988) *Self-Organization and Associative Memory*, 2nd Ed., Springer Verlag